# Adaptive Learning of Smoothing Functions: Application to Electricity Load Forecasting

**Amadou Ba**
IBM Research - Ireland
Mulhuddart, Dublin 15
amadouba@ie.ibm.com

**Mathieu Sinn**
IBM Research - Ireland
Mulhuddart, Dublin 15
mathsinn@ie.ibm.com

**Yannig Goude**
EDF R&D
Clamart, France
yannig.goude@edf.fr

**Pascal Pompey**
IBM Research - Ireland
Mulhuddart, Dublin 15
papompey@ie.ibm.com

## Abstract

This paper proposes an efficient online learning algorithm to track the smoothing functions of Additive Models. The key idea is to combine the linear representation of Additive Models with a Recursive Least Squares (RLS) filter. In order to quickly track changes in the model and put more weight on recent data, the RLS filter uses a forgetting factor which exponentially weights down observations by the order of their arrival. The tracking behaviour is further enhanced by using an adaptive forgetting factor which is updated based on the gradient of the a priori errors. Using results from Lyapunov stability theory, upper bounds for the learning rate are analyzed. The proposed algorithm is applied to 5 years of electricity load data provided by the French utility company Electricité de France (EDF). Compared to state-of-the-art methods, it achieves a superior performance in terms of model tracking and prediction accuracy.

## 1 Introduction

Additive Models are a class of nonparametric regression methods which have been the subject of intensive theoretical research and found widespread applications in practice (see [1]). This considerable attention comes from the ability of Additive Models to represent non-linear associations between covariates and response variables in an intuitive way, and the availability of efficient training methods. The fundamental assumption of Additive Models is that the effect of covariates on the dependent variable follows an additive form. The separate effects are modeled by smoothing splines, which can be learned using penalized least squares.

A particularly fruitful field for the application of Additive Models is the modeling and forecasting of short term electricity load. There exists a vast body of literature on this subject, covering methods from statistics (Seasonal ARIMA models [2, 3], Exponential Smoothing [4], regression models [5, 6, 7]) and, more recently, also from machine learning [8, 9, 10]. Additive Models were applied, with good results, to the nation-wide load in France [11] and to regional loads in Australia [12]. Besides electricity load, Additive Models have also been applied to natural gas demand [13].

Several methods have been proposed to track time-varying behaviour of the smoothing splines in Additive Models. Hoover et al. [14] examine estimators based on locally weighted polynomials and derive some of their asymptotic properties. In a similar vein, Eubank et al. [15] introduce a Bayesian approach which can handle multiple responses. A componentwise smoothing spline is suggested by Chiang et al. [16]. Fan and Zhang [17] propose a two-stage algorithm which first computes raw

estimates of the smoothing functions at different time points and then smoothes the estimates. A comprehensive review can be found in [18]. A common feature of all these methods is that they identify and estimate the time-varying behaviour *a posteriori*.

Adaptive learning of Additive Models in an *online* fashion is a relatively new topic. In [19], an algorithm based on iterative QR decompositions is proposed, which yields promising results for the French electricity load but also highlights the need for a forgetting factor to be more reactive, e.g., to macroeconomic and meteorological changes, or varying consumer portfolios. Harvey and Koopman [20] propose an adaptive learning method which is restricted to changing periodic patterns. Adaptive methods of a similar type have been studied in the field of neural networks [21, 22].

The contributions of our paper are threefold: First, we introduce a new algorithm which combines Additive Models with a Recursive Least Squares (RLS) filter to track time-varying behaviour of the smoothing splines. Second, in order to enhance the tracking ability, we consider filters that include a forgetting factor which can be either fixed, or updapted using a gradient descent approach [23]. The basic idea is to decrease the forgetting factor (and hence increase the reactivity) in transient phases, and increasing the forgetting factor (thus decreasing the variability) during stationary regimes. Using results from Lyapunov stability theory [24], we provide a theoretical analysis of the learning rate in the gradient descent approach. Third, we evaluate the proposed methodology on 5 years of electricity load data provided by the French utility company Electricité de France (EDF). The results show that the adaptive learning algorithm outperforms state-of-the-art methods in terms of model tracking and prediction accuracy. Moreover, the experiments demonstrate that using an adaptive forgetting factor stabilizes the algorithm and yields results comparable to those obtained by using the (*a priori* unknown) optimal value for a fixed forgetting factor. Note that, in this paper, we do not compare our proposed algorithm with existing online learning methods from the machine learning literature, such as tracking of best experts (see [25] for an overview). The reason is that we are specifically interested in adaptive versions of Additive Models, which have been shown to be particularly well-suited for modeling and forecasting electricity load.

The remainder of the paper is organized as follows. Section 2 reviews the definition of Additive Models and provides some background on the spline representation of smoothing functions. In Section 3 we present our adaptive learning algorithms which combine Additive Models with a Recursive Least Squares (RLS) filter. We discuss different approaches for including forgetting factors and analyze the learning rate for the gradient descent method in the adaptive forgetting factor approach. A case study with real electricity load data from EDF is presented in Section 4. An outlook on problems for future research concludes the paper.

## 2  Additive Models

In this section we review the Additive Models and provide background information on the spline representation of smoothing functions. Additive Models have the following form:

$$y_k \;=\; \sum_{i=1}^{I} f_i(x_k) + \epsilon_k.$$

In this formulation, $x_k$ is a vector of covariates which can be either categorical or continuous, and $y_k$ is the dependent variable, which is assumed to be continuous. The noise term $\epsilon_k$ is assumed to be Gaussian, independent and identically distributed with mean zero and finite variance. The functions $f_i$ are the *transfer functions* of the model, which can be of the following types: constant (exactly one transfer function, representing the intercept of the model), categorical (evaluating to $0$ or $1$ depending on whether the covariates satisfy certain conditions), or continuous. The continuous transfer functions can be either linear functions of covariates (representing simple linear trends), or smoothing splines. Typically, smoothing splines depend on only 1-2 of the continuous covariates. An interesting possibility is to combine smoothing splines with categorical conditions; in the context of electricity load modeling this allows, e.g., for having different effects of the time of the day depending on the day of the week.

In our experiments, we use 1- and 2-dimensional cubic B-splines, which allows us to write the smoothing splines in the following form:

$$f_i(x_k) \;=\; \boldsymbol{\beta}_i^T \boldsymbol{b}_i(x_k) \;=\; \sum_{j=1}^{J_i} \beta_{ij} b_{ij}(x_k), \tag{1}$$

where $\beta_{ij}$ are the spline coefficients and $b_{ij}$ are the spline basis functions which depend on 1 or 2 components of $x_k$. Note that the basis functions are defined by a (fixed) sequence of knot points, while the coefficients are used to fit the spline to the data (see [1] for details). The quantity $J_i$ in equation (1) is the number of spline coefficients associated with the transfer function $f_i$. Now, let $\boldsymbol{\beta}$ denote the stacked vector containing the spline coefficients, and $\boldsymbol{b}(x_k)$ the stacked vector containing the spline basis functions of all the transfer functions. This allows us to write the Additive Models in the following linear form:

$$y_k \;=\; \boldsymbol{\beta}^T \boldsymbol{b}(x_k) + \epsilon_k. \tag{2}$$

## 2.1 Learning Additive Models

The linear representation of Additive Models in (2) is the starting point for efficient learning algorithms. Consider $K$ samples $(x_k, y_k)$, $k = 1, \ldots, K$ of covariates and dependent variables. Then an estimate of the model coefficients $\boldsymbol{\beta}$ can be obtained by solving the following *weighted penalized least squares* problem:

$$\widehat{\boldsymbol{\beta}}_K = \min_{\boldsymbol{\beta}} \left\{ (\boldsymbol{y}_K - \boldsymbol{B}_K \boldsymbol{\beta})^T \, \boldsymbol{\Omega}_K \, (\boldsymbol{y}_K - \boldsymbol{B}_K \boldsymbol{\beta}) + \boldsymbol{\beta}^T \boldsymbol{S}_K \boldsymbol{\beta} \right\}. \tag{3}$$

Here $\boldsymbol{y}_K = (y_1, y_2, \ldots, y_K)^T$ is the $K \times 1$ vector containing all the dependent variables, $\boldsymbol{B}_K$ is the matrix with the rows $\boldsymbol{b}(x_1)^T$, $\boldsymbol{b}(x_2)^T$, $\ldots$, $\boldsymbol{b}(x_K)^T$ containing the evaluated spline basis functions. The matrix $\boldsymbol{\Omega}_K$ puts different weights on the samples. In this paper, we consider two scenarios: $\boldsymbol{\Omega}_K$ is the identity matrix (putting equal weight on the $K$ regressors), or a diagonal matrix which puts exponentially decreasing weights on the samples, according to the order of their arrival (thus giving rise to the notion of *forgetting factors*). The different weighting schemes are discussed in more detail in Section 3. The matrix $\boldsymbol{S}_K$ in (3) introduces a penalizing term in order to avoid overfitting of the smoothing splines. In this paper, we use diagonal penalizers not depending on the sample size $K$:

$$\boldsymbol{S} \;=\; \mathrm{diag}(\gamma, \gamma, \ldots, \gamma), \tag{4}$$

where $\gamma > 0$. Note that this penalizer shrinks the smoothing splines towards zero functions, and the strength of this effect is tuned by $\gamma$. As a well-known fact (see [1]), provided that the matrix $(\boldsymbol{B}_K^T \boldsymbol{\Omega}_K \boldsymbol{B}_K + \boldsymbol{S})$ is non-singular, the above least squares problem has the closed-form solution

$$\widehat{\boldsymbol{\beta}}_K \;=\; (\boldsymbol{B}_K^T \boldsymbol{\Omega}_K \boldsymbol{B}_K + \boldsymbol{S})^{-1} \boldsymbol{B}_K^T \boldsymbol{\Omega}_K \boldsymbol{y}_K. \tag{5}$$

## 3 Adaptive learning of smoothing functions

Equation (5) gives rise to an efficient batch learning algorithm for Additive Models. Next, we propose an adaptive method which allows us to track changes in the smoothing functions in an online fashion. The basic idea is to combine the linear representation of Additive Models in (2) with classical Recursive Least Squares (RLS) filters. To improve the tracking behaviour, we introduce a *forgetting factor* which puts more weight on recent samples. See Algorithm 1 for details. As starting values, we choose $\widehat{\boldsymbol{\beta}}_0$ equal to an initial estimate of $\boldsymbol{\beta}$ (e.g., obtained in previous experiments), or equal to a zero vector if no prior information is available. The initial precision matrix $\boldsymbol{P}_0$ is set equal to the inverse of the penalizer $\boldsymbol{S}$ in (4). Anytime while the algorithm is running, the current estimate $\widehat{\boldsymbol{\beta}}_k$ can be used to compute predictions for new given covariates.

Let us discuss the role of the forgetting factor $\omega$ in the adaptive learning algorithm. First, note that Algorithm 1 is equivalent to the solution of the weighted least squares problem in (5) with the weighting matrix $\boldsymbol{\Omega}_K = \mathrm{diag}(\omega^{K-1}, \omega^{K-2}, \ldots, \omega^2, \omega, 1)$ and the penalizer $\boldsymbol{S}$ as defined in (4). If $\omega = 1$, all samples are weighted equally. For $\omega < 1$, samples are discounted exponentially according to the order of their arrival. In general, a smaller forgetting factor improves the tracking of temporal changes in the model coefficients $\boldsymbol{\beta}$. This reduction of the bias typically comes at the cost of an increase of the variance. Therefore, finding the right balance between the forgetting factor $\omega$ and the strength $\gamma$ of the penalizer in (4) is crucial for a good performance of the forecasting algorithm.

---

**Algorithm 1** Adaptive learning (fixed forgetting factor)

---

1: **Input:** Initial estimate $\widehat{\boldsymbol{\beta}}_0$, forgetting factor $\omega \in (0, 1]$, penalizer strength $\gamma > 0$.
2: Compute the initial precision matrix $\boldsymbol{P}_0 = \mathrm{diag}(\gamma^{-1}, \gamma^{-1}, \dots, \gamma^{-1})$.
3: **for** $k = 1, 2, \dots$ **do**
4:     Obtain new covariates $x_k$ and dependent variable $y_k$.
5:     Compute the spline basis functions $\boldsymbol{b}_k = \boldsymbol{b}(x_k)$.
6:     Compute the *a priori error* and the *Kalman gain*:

$$\widehat{\epsilon}_k = y_k - \boldsymbol{b}_k^T \widehat{\boldsymbol{\beta}}_{k-1},$$
$$\boldsymbol{g}_k = \frac{\boldsymbol{P}_{k-1} \boldsymbol{b}_k}{\omega + \boldsymbol{b}_k^T \boldsymbol{P}_{k-1} \boldsymbol{b}_k}.$$

7:     Update the estimate and the precision matrix:

$$\widehat{\boldsymbol{\beta}}_k = \widehat{\boldsymbol{\beta}}_{k-1} + \boldsymbol{g}_k \widehat{\epsilon}_k,$$
$$\boldsymbol{P}_k = \omega^{-1} \left[ \boldsymbol{P}_{k-1} - \boldsymbol{g}_k \boldsymbol{b}_k^T \boldsymbol{P}_{k-1} \right].$$

8: **end for**

---

---

**Algorithm 2** Adaptive learning (adaptive forgetting factor)

---

1: **Input:** Initial estimate $\widehat{\boldsymbol{\beta}}_0$, initial forgetting factor $\omega_0 \in (0, 1]$, lower bound for the forgetting factor $\omega_{\min} \in (0, 1]$, learning rate $\eta > 0$, penalizer strength $\gamma > 0$.
2: Same as Step 2 in Algorithm 1.
3: Set $\boldsymbol{\psi}_0$ equal to a zero vector and $\boldsymbol{\Psi}_0$ to the identity matrix.
4: **for** $k = 1, 2, \dots$ **do**
5:     Same as Steps 4-6 in Algorithm 1, with $\omega_{k-1}$ instead of $\omega$.
6:     Update the forgetting factor:

$$\omega_k = \omega_{k-1} + \eta \, \boldsymbol{b}_k^T \boldsymbol{\psi}_{k-1} \widehat{\epsilon}_k.$$

    If $\omega_k > 1$, then set $\omega_k$ equal to 1. If $\omega_k < \omega_{\min}$, then set $\omega_k$ equal to $\omega_{\min}$.
7:     Same as Step 7 in Algorithm 1, with $\omega_k$ instead of $\omega$.
8:     Compute the updates (where $\boldsymbol{I}$ denotes the identity matrix):

$$\boldsymbol{\Psi}_k = \omega_k^{-1} \left( \boldsymbol{I} - \boldsymbol{g}_k \boldsymbol{b}_k^T \right) \boldsymbol{\Psi}_{k-1} \left( \boldsymbol{I} - \boldsymbol{b}_k \boldsymbol{g}_k^T \right) - \omega_k^{-1} \boldsymbol{P}_k + \omega_k^{-1} \boldsymbol{g}_k \boldsymbol{g}_k^T,$$
$$\boldsymbol{\psi}_k = \left( \boldsymbol{I} - \boldsymbol{g}_k \boldsymbol{b}_k^T \right) \boldsymbol{\psi}_{k-1} + \boldsymbol{\Psi}_k \boldsymbol{b}_k \widehat{\epsilon}_k.$$

9: **end for**

---

## 3.1 Adaptive forgetting factors

In this section we present a modification of Algorithm 1 which uses adaptive forgetting factors in order to improve the stability and the tracking behaviour. The basic idea is to choose a large forgetting factor during stationary regimes (when the *a priori* errors are small), and small forgetting factors during transient phases (when the *a priori* error is large). In this paper we adopt the gradient descent approach in [23] and update the forgetting factor according to the following formula:

$$\omega_k = \omega_{k-1} - \eta \frac{\partial \mathbb{E}[\widehat{\epsilon}_k^2]}{\partial \omega_{k-1}}.$$

Searching in the direction of the partial derivative of $\mathbb{E}[\widehat{\epsilon}_k^2]$ with respect to $\omega_{k-1}$ aims at minimizing the expected value of the *a priori* errors. The *learning rate* $\eta > 0$ determines the reactivity of the algorithm: if it is high, then the errors lead to large decreases of the forgetting factor, and vice versa. The details of the adaptive forgetting factor approach are given in Algorithm 2.

Note that $\omega_k$ is updated in an iterative fashion based on $\boldsymbol{\psi}_k$ (the gradient of the estimate $\widehat{\boldsymbol{\beta}}_k$ with respect to $\omega_{k-1}$), and on $\boldsymbol{\Psi}_k$ (the gradient of the precision matrix $\boldsymbol{P}_k$ with respect to $\omega_{k-1}$).

## 3.2 Stability analysis

In the following, we apply results from Lyapunov stability theory to analyze the effect of the learning rate $\eta$. We show how to derive analytical bounds for $\eta$ that guarantee stability of the algorithm.

Recall the definition of the *a priori* error, $\widehat{\epsilon}_k = y_k - \boldsymbol{b}_k^T \widehat{\boldsymbol{\beta}}_{k-1}$. As equilibrium point of our algorithm, we consider the ideal situation $\widehat{\epsilon}_k = 0$. We choose the candidate Lyapunov function $V(\widehat{\epsilon}_k) = \widehat{\epsilon}_k^2/2$. Clearly, the following conditions are satisfied: if $x = 0$ then $V(x) = 0$; if $x \neq 0$ then $V(x) > 0$; and $V(x) \to \infty$ as $x \to \infty$. Consider the discrete time derivative $\Delta V(\widehat{\epsilon}_k) = V(\widehat{\epsilon}_{k+1}) - V(\widehat{\epsilon}_k)$ of the candidate Lyapunov function. According to Lyapunov stability theory, if $V(\widehat{\epsilon}_k) > 0$ and $\Delta V(\widehat{\epsilon}_k) < 0$, then $V(\widehat{\epsilon}_k)$ converges to zero as $k$ tends to infinity.

Let us analyze $\Delta V(\widehat{\epsilon}_k)$ more closely. Using the relation $\widehat{\epsilon}_k = \Delta \widehat{\epsilon}_{k+1} + \widehat{\epsilon}_k$ we arrive at

$$\Delta V(\widehat{\epsilon}_k) = \Delta \widehat{\epsilon}_k \left( \frac{1}{2} \Delta \widehat{\epsilon}_k + \widehat{\epsilon}_k \right). \tag{6}$$

Next we approximate $\Delta \widehat{\epsilon}_k$ by its first order Taylor series expansion:

$$\Delta \widehat{\epsilon}_k = \frac{\partial \widehat{\epsilon}_k}{\partial \omega_k} \Delta \omega_k. \tag{7}$$

Furthermore, note that

$$\frac{\partial \widehat{\epsilon}_k}{\partial \omega_k} = -\boldsymbol{b}_k^T \boldsymbol{\psi}_{k-1} \quad \text{and} \quad \Delta \omega_k = \eta \widehat{\epsilon}_k \boldsymbol{b}_k^T \boldsymbol{\psi}_{k-1}. \tag{8}$$

Substituting the expressions in (7) and (8) back into (6), we obtain the approximation

$$\Delta V(\widehat{\epsilon}_k) = \left( -\boldsymbol{b}_k^T \boldsymbol{\psi}_{k-1} \right) \left( \eta \widehat{\epsilon}_k \boldsymbol{b}_k^T \boldsymbol{\psi}_{k-1} \right) \left[ \frac{1}{2} \left( -\boldsymbol{b}_k^T \boldsymbol{\psi}_{k-1} \right) \left( \eta \widehat{\epsilon}_k \boldsymbol{b}_k^T \boldsymbol{\psi}_{k-1} \right) + \widehat{\epsilon}_k \right].$$

After some basic algebraic manipulations we arrive at the approximation

$$\Delta V(\widehat{\epsilon}_k) = \frac{1}{2} \eta \widehat{\epsilon}_k^2 \left( \boldsymbol{b}_k^T \boldsymbol{\psi}_{k-1} \right)^2 \left( -2 + \eta \left( \boldsymbol{b}_k^T \boldsymbol{\psi}_{k-1} \right)^2 \right). \tag{9}$$

Now it is easy to see that an (approximate) equivalent condition for Lyapunov stability is given by

$$0 < \eta < \frac{2}{\left( \boldsymbol{b}_k^T \boldsymbol{\psi}_{k-1} \right)^2}.$$

## 4 Case study: Forecasting of electricity load

In this section, we apply our adaptive learning algorithms to real electricity load data provided by the French utility company Electricité de France (EDF). Modeling and forecasting electricity load is a challenging task due to the non-linear effects, e.g., of the temperature and the time of the day. Moreover, the electricity load exhibits many non-stationary patterns, e.g., due to changing macroeconomic conditions (leading to an increase/decrease in electricity demand), or varying customer portfolios resulting from the liberalization of European electricity markets. The performance on these highly complex, non-linear and non-stationary learning tasks is a challenging benchmark for our adaptive algorithms.

### 4.1 Experimental data

The dependent variables $y_k$ in the data provided by EDF represent half-hourly electricity load measurements between February 2, 2006 and April 6, 2011. The covariates $x_k$ include the following information:

$$x_k = \left( x_k^{\text{DayType}}, x_k^{\text{TimeOfDay}}, x_k^{\text{TimeOfYear}}, x_k^{\text{Temperature}}, x_k^{\text{CloudCover}}, x_k^{\text{LoadDecrease}} \right).$$

Let us explain these components in more detail:

- $x_k^{\text{DayType}}$ is a categorical variable representing the day type: 1 for Sunday, 2 for Monday, 3 for Tuesday-Wednesday-Thursday, 4 for Friday, 5 for Saturday, and 6 for bank holidays.

- $x_k^{\text{TimeOfDay}}$ is the index (in half-hourly time steps) of the current time within the day. Its values range from 0 for 0.00 am to 47 for 11.30 pm.

- $x_k^{\text{TimeOfYear}}$ is the position of the current day within the year (taking values from 0 for January 1, to 1 for December 31).

- $x_k^{\text{Temperature}}$ and $x_k^{\text{CloudCover}}$ represent the temperature and the cloud cover (ranging from 0 for a blue sky to 8 for overcast). These meteorological covariates have been provided by Météo France; the raw data include temperature and cloud cover data recorded every 3 hours from 26 weather stations all over France. We interpolate these measurements to obtain half-hourly data. A weighted average – the weights reflecting the importance of a region in terms of the national electricity load – is computed to obtain the national temperature and cloud cover covariates.

- $x_k^{\text{LoadDecrease}}$ contains information about the activation of contracts between EDF and some big customers to reduce the electricity load during peak days.

We partition the data into two sets: a *training set* from February 2, 2006 to August 31, 2010, and a *test set* from September 1, 2010 to April 6, 2011.

## 4.2 Modeling the electricity load

We use the following Additive Model for the electricity load:

$$
\begin{aligned}
y_k \;=\;\; & \beta^{\text{Intercept}} \;+\; f^{\text{Trend}}(k) \;+\; f^{\text{LagLoad}}(y_{k-48}) \;+\; \sum_{l=1}^{6} \mathbf{1}(x_k^{\text{DayType}} = l)(\beta_l^{\text{DayType}} + f_l^{\text{TimeOfDay}}(x_k)) \\
+\;\; & f^{\text{CloudCover}}(x_k) \;+\; f^{\text{Temperature/TimeOfDay}}(x_k) \;+\; f^{\text{LagTemperature}}(x_{k-48}) \\
+\;\; & f^{\text{TimeOfYear}}(x_k) \;+\; x_k^{\text{LoadDecrease}}\, f^{\text{LoadDecrease}}(x_k) \;+\; \epsilon_k.
\end{aligned}
$$

Let us explain the model in more detail:

- The intercept $\beta^{\text{Intercept}}$ models the base load, and $f^{\text{Trend}}(k)$ captures non-linear trends, e.g., due to the economic crisis and changes in the customer portfolios of EDF.

- $f^{\text{LagLoad}}(y_{k-48})$ takes into account the electricity load of the previous day.

- $\beta_l^{\text{DayType}}$ and $f_l^{\text{TimeOfDay}}(x_k)$ capture the day-type specific effects of the time of the day.

- $f^{\text{CloudCover}}(x_k)$ and $f^{\text{Temperature/TimeOfDay}}(x_k)$ represent respectively the effect of the cloud cover and the bivariate effect of the temperature and the time of the day.

- The term $f^{\text{LagTemperature}}(x_{k-48})$ takes into account the temperature of the previous day, which is important to capture the thermal inertia of buildings.

- $f^{\text{TimeOfYear}}(x_k)$ represents yearly cycles, and $x_k^{\text{LoadDecrease}}\, f^{\text{LoadDecrease}}(x_k)$ models the effect of contracts to reduce peak loads depending on the time of the day.

To fit the model we use the R package `mgcv` (see [26, 27]). For more information about the design of models for electricity data we refer to [19, 11]. Figure 1 shows the estimated joint effect of the temperature and the time of the day, and the estimated yearly cycle. As to be expected, low (resp. high) temperatures lead to an increase of the electricity load due to electrical heating (resp. cooling), whereas temperatures between $10°$ and $20°$ Celsius have almost no effect on the electricity load. Due to the widespread usage of electrical heating and relatively low usage of air conditioning in France, the effect of heating is approximately four times higher than the effect of cooling. The yearly cycle reveals a strong decrease of the electricity load during the summer and Christmas holidays (around 0.6 and 1 of the time of the year). Note that the scales of the effects have been normalized because of data confidentiality reasons.

The fitted model achieves a good performance on the training data set with an adjusted R-square of 0.993, a Mean Absolute Percentage Error (MAPE) of $1.4\%$, and a Root Mean Square Error (RMSE) of 835 MW. All the incorporated effects yield significant improvements in terms of the Generalized Cross Validation (GCV) score, so the model size cannot be reduced. The fitted model consists of 268 spline basis coefficients, which indicates the complexity of modeling electricity load data.

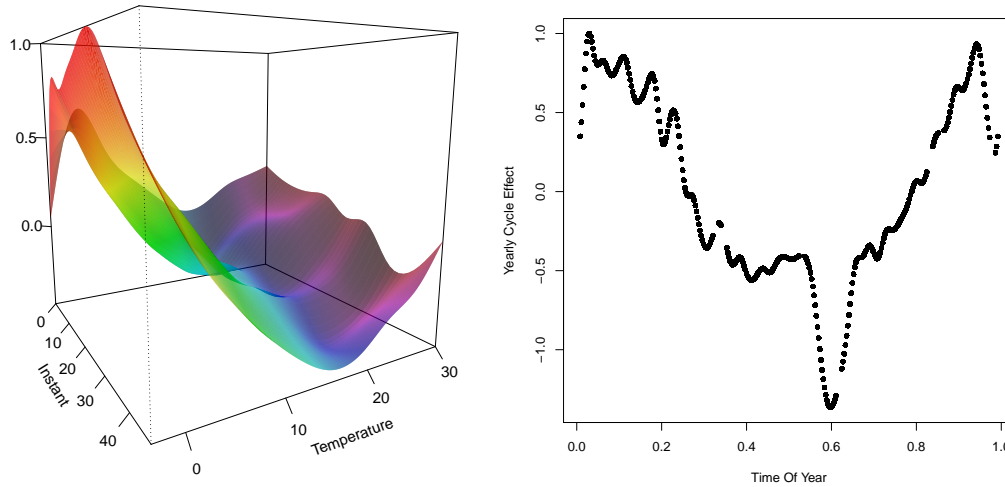

Figure 1: Effect of the temperature and the time of the day (left), and yearly cycle (right).

## 4.3 Adaptive learning and forecasting

We compare the performance of five different algorithms:

- The offline method (denoted by **ofl**) uses the model learned in R and applies it to the test data without updating the model parameters.

- The fixed forgetting factor method (denoted by **fff**) updates the Additive Model using a fixed forgetting factor (see Algorithm 1). The value of the fixed forgetting factor and the strength of the penalizer are determined in the following way: We divide the test set into two parts of equal length, a *calibration set* (September 1, 2010 - November 15, 2010) and a *validation set* (November 16, 2010 - April 6, 2011). We choose the combination of forgetting factor and penalizer strength which yields the best results on the calibration set in terms of MAPE and RMSE, and evalute the performance on the validation set.

- The post-fixed forgetting factor method (denoted by **post-fff**) uses the fixed forgetting factor and strength of the penalizer which yield the best performance on the validation set. This "ideal" parameterization gives us an upper bound for the performance of the fff method and a benchmark for the adaptive forgetting factor approaches.

- The adaptive forgetting factor method (denoted by **aff**) uses Algorithm 2.

- Finally, we evaluate an adaptive approach that optimizes the values of the forgetting factor and the penalizer strength on a grid (denoted by **affg**): For each combination on the grids $(0.995, 0.996, ..., 0.999)$ and $(1000, 2000, ..., 10000)$, we run fixed forgetting factor algorithms in parallel. At each time point, we choose the combination which so far has given the best performance in terms of MAPE.

## 4.4 Results

The performance of all five algorithms is evaluted on the validation set from November 16, 2010 to April 6, 2011. Table 1 shows the results in terms of MAPE and RMSE. As can be seen, the adaptive forgetting factor method (aff) achieves the best performance. It even outperforms the post-fff method which uses the (*a priori* unknown) optimal combination of penalizer strength and fixed forgetting factor. The improvements over the offline approach (which doesn't update the model parameters) are significant both in terms of the MAPE (about 0.2%) and the RMSE (about 100 MW). This corresponds to an improvement of approximatively 10% in terms of the day-ahead forecasting error.

Figure 2 (left) shows the cumulative sum of the errors of the five forecasting algorithms. As can be seen, the offline approach suffers from a strong positive bias and tends to overestimate the electricity load over time. In fact, there was a decrease in the electricity demand over the considered time horizon due to the economic crisis. The adaptive forgetting factor shows a much better tracking behaviour and is able to adapt to the change in the demand patterns.

The graph on the right hand side of Figure 2 illustrates the roles of the forgetting factor and of the strength of the penalizer. Values of the forgetting factor close to 1 result in reduced tracking behaviour and less improvement over the offline approach. Choosing too small values for the forgetting factor can lead to loss of information and instabilities of the algorithm. Increasing the penalizer reduces the variability of the smoothing splines, however, it also introduces a bias as the splines are shrinked towards zero.

| Method | ofl | fff | affg | aff | post-fff |
|---|---|---|---|---|---|
| MAPE (%) | 1.83 | 2.28 | 1.7 | 1.63 | 1.64 |
| RMSE (MW) | 1185 | 1869 | 1124 | 1071 | 1073 |

Table 1: Performance of the five different forecasting methods

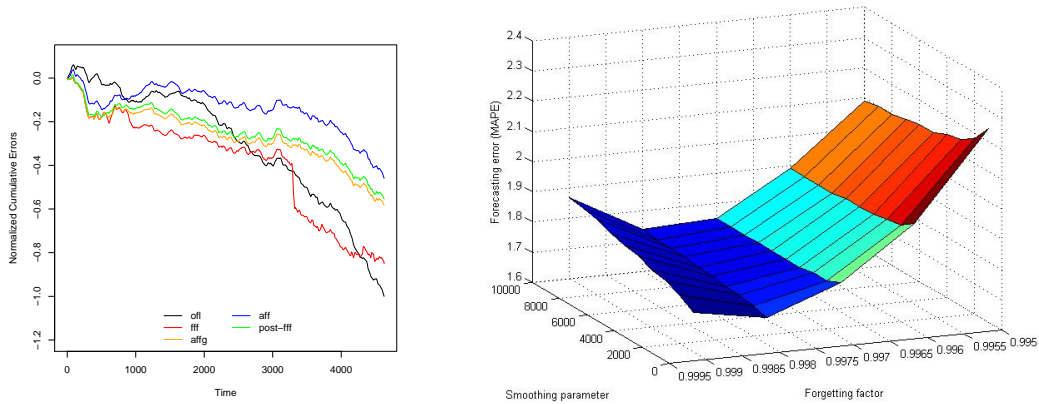

Figure 2: Cumulative sum of the errors (left) and results for different choices of the forgetting factor and the strength of the penalizer (right)

## 5   Conclusions and future work

We have presented an adaptive learning algorithm that updates the smoothing functions of Additive Models in an online fashion. We have introduced methods to improve the tracking behaviour based on forgetting factors and analyzed theoretical properties using results from Lyapunov stability theory. The significance of the proposed algorithms was demonstrated in the context of forecasting electricity load data. Modeling and forecasting electricity load data is particularly challenging due to the high complexity of the models (the Additive Models in our experiments included 268 spline basis functions), the non-linear relation between the covariates and dependent variables, and the non-stationary dynamics of the models. Experiments on 5 years of data from Electricité de France have shown the superior performance of algorithms using an adaptive forgetting factor. As it turned out, a crucial point is to find the right combination of forgetting factors and the strength of the penalizer. While forgetting factors tend to reduce the bias of models evolving over time, they typically increase the variance, an effect which can be compensated by choosing stronger penalizer. Our future research will follow two directions: first, we plan to consider dynamic penalizers which can automatically adapt to changes in the model complexity. Second, we will develop methods for incorporating prior information on model components, e.g., by integrating beliefs for the initial values of the adaptive algorithms.

## References

[1] Trevor Hastie, Robert Tibshirani, and Jerome Friedman. *The Elements of Statistical Learning*. Second Edition, Springer, 2009.

[2] J Nowicka-Zagrajek and R Weron. Modeling electricity loads in California: ARMA models with hyperbolic noise. *Signal Processing*, pages 1903–1915, 2002.

[3] Shyh-Jier Huang and Kuang-Rong Shih. Short-Term Load Forecasting Via ARMA Model Identification Including Non-Gaussian Process Considerations. *IEEE Transactions on Power Systems*, 18(2):673–679, 2003.

[4] James W. Taylor. Short-Term Load Forecasting with Exponentially Weighted Methods. *IEEE Transactions on Power Systems*, 27(1):673–679, 2012.

[5] Derek W. Bunn and E. D. Farmer. *Comparative models for electrical load forecasting*. Eds. Wiley, New York, 1985.

[6] R Campo and P. Ruiz. Adaptive Weather-Sensitive Short Term Load Forecast . *IEEE Transactions on Power Systems*, 2(3):592–598, 1987.

[7] Ramu Ramanathan, Robert Engle, Clive W. J. Granger, Farshid Vahid-Araghi, and Casey Brace. Short-run forecasts of electricity loads and peaks . *International Journal of Forecasting*, 13(3):161–174, 1997.

[8] Bo-Juen Chen, Ming-Wei Chang, and Chih-Jen Lin. Load Forecasting Using Support Vector Machines: A Study on EUNITE Competition 2001 . *IEEE Transaction on Power Systems*, 19(3):1821–1830, 2004.

[9] Shu Fan and Luonan Chen. Short-term load forecasting based on an adaptive hybrid method . *IEEE Transaction on Power Systems*, 21(1):392–401, 2006.

[10] V. H Hinojosa and A Hoese. Short-Term Load Forecasting Using Fuzzy Inductive Reasoning and Evolutionary Algorithms . *IEEE Transaction on Power Systems*, 25(1):565–574, 2010.

[11] A Pierrot and Yannig Goude. Short-term electricity load forecasting with generalized additive models. In *Proceedings of ISAP Power*, pages 593–600, 2011.

[12] Shu Fan and R Hyndman. Short-Term Load Forecasting Based on a Semi-Parematetric Additive Model . *IEEE Transaction on Power Systems*, 27(1):134–141, 2012.

[13] M Brabek, O Konr, M Mal, M Pelikn, and J Vondrcek. A statistical model for natural gas standardized load profiles. *Journal of the Royal Statistical Society: Series C (Applied Statistics)*, 58(1):123–139, 2009.

[14] Donald R. Hoover, John A. Rice, Colin O. Wu, and Li-Ping Yang. Nonparametric smoothing estimates of time-varying coefficient models with longitudinal data. *Biometrika*, 85(4):809–822, 1998.

[15] R. L. Eubank, Chunfeng Huang, Y. Munoz. Maldonado, and R. J. Buchanan. Smoothing spline estimation in varying coefficient models. *Journal of the Royal Statistical Society*, 66(3):653–667, 2004.

[16] Chin-Tsang Chiang, John A. Rice, and Colin O. Wu. Smoothing spline estimation for varying coefficient models with repeatedly measured dependent variables. *Journal of the American Statistical Asociation*, 96(454):605–619, 2001.

[17] Jianqing Fan and Jin-Ting Zhang. Two-Step Estimation of Functional Linear Models with Applications to Longitudinal Data. *Journal of the Royal Statistical Society*, 62:303–322, 2000.

[18] Jianqing Fan and Wenyang Zhang. Statistical methods with varying coefficient models. *Statistics and Its Interface*, 1:179–195, 2008.

[19] S. Wood, Y. Goude, and S. Shaw. *Generalized Additive Models*. Preprint, 2011.

[20] A Harvey and S. J Koopman. Forecasting Hourly Electricity Demand Using Time-Varying Splines. *Journal of the American Statistical Association*, 88(424):1228–1236, 1993.

[21] Herbert Jaeger. Adaptive non-linear system identification with echo state networks. In *Proc. Advances Neural Information Processing Systems*, pages 593–600, 2002.

[22] Mauro Birattari, Gianluca Bontempi, and Hugues Bersini. Lazy learning meets the recursive least squares algorithm. In *Proc. Advances Neural Information Processing Systems*, pages 375–381, 1999.

[23] S-H Leung and C. F So. Gradient-Based Variable Forgetting Factor RLS Algorithm in Time-Varying Environments. *IEEE Transaction on Signal Processing*, 53(8):3141–3150, 2005.

[24] Z Man, H. R Wu, S Liu, and X Yu. A New Adaptive Backpropagation Algorithm Based on Lyapunov Stability Theory for Neural Network. *IEEE Transaction on Neural Networks*, 17(6):1580–1591, 2006.

[25] Nicolò Cesa-Bianchi and Gábor Lugosi. *Prediction, Learning, and Games*. Cambridge University Press, 2006.

[26] Simon Wood. *Generalized Additive Models an Introduction with R*. Chapman and Hall Eds., 2006.

[27] Simon Wood. mgcv :GAMs and Generalized Ridge Regression for R. *R News*, 1(2):20–25, 2001.
